# Estimating Robust Query Models
# with Convex Optimization

**Kevyn Collins-Thompson**[*]
Microsoft Research
1 Microsoft Way
Redmond, WA U.S.A. 98052
`kevynct@microsoft.com`

## Abstract

Query expansion is a long-studied approach for improving retrieval effectiveness by enhancing the user's original query with additional related words. Current algorithms for automatic query expansion can often improve retrieval accuracy on average, but are not *robust*: that is, they are highly unstable and have poor worst-case performance for individual queries. To address this problem, we introduce a novel formulation of query expansion as a convex optimization problem over a word graph. The model combines initial weights from a baseline feedback algorithm with edge weights based on word similarity, and integrates simple constraints to enforce set-based criteria such as aspect balance, aspect coverage, and term centrality. Results across multiple standard test collections show consistent and significant reductions in the number and magnitude of expansion failures, while retaining the strong positive gains of the baseline algorithm. Our approach does not assume a particular retrieval model, making it applicable to a broad class of existing expansion algorithms.

## 1 Introduction

A major goal of current information retrieval research is to develop algorithms that can improve retrieval effectiveness by inferring a more complete picture of the user's information need, beyond that provided by the user's query text. A *query model* captures a richer representation of the context and goals of a particular information need. For example, in the language modeling approach to retrieval [9], a simple query model may be a unigram language model, with higher probability given to terms related to the query text. Once estimated, a query model may be used for such tasks as query expansion, suggesting alternate query terms to the user, or personalizing search results [11]. In this paper, we focus on the problem of automatically inferring a query model from the top-ranked documents obtained from an initial query. This task is known as pseudo-relevance feedback or blind feedback, because we do not assume any direct input from the user other than the initial query text. Despite decades of research, even state-of-the-art methods for inferring query models – and in particular, pseudo-relevance feedback – still suffer from some serious drawbacks. First, past research efforts have focused largely on achieving good average performance, without regard for the *stability* of individual retrieval results. The result is that current models are highly unstable and have bad worst-case performance for individual queries. This is one significant reason that Web search engines still make little or no use of automatic feedback methods. In addition, current methods do not

---

[*]This work was primarily done while the author was at the Language Technologies Institute, School of Computer Science, Carnegie Mellon University.

adequately capture the relationships or tradeoffs between competing objectives, such as maximizing the expected relevance weights of selected words versus the risks of those choices. This is turn leads to several problems.

First, when term risk is ignored, the result will be less reliable algorithms for query models, as we show in Section 3. Second, selection of expansion terms is typically done in a greedy fashion by rank or score, which ignores the properties of the terms *as a set* and leads to the problem of aspect imbalance, a major source of retrieval failures [2]. Third, few existing expansion algorithms can operate *selectively*; that is, automatically detect when a query is risky to expand, and then avoid or reduce expansion in such cases. The few algorithms we have seen that do attempt selective expansion are not especially effective, and rely on sometimes complex heuristics that are integrated in a way that is not easy to untangle, modify or refine. Finally, for a given task there may be additional factors that must be constrained, such as the computational cost of sending many expansion terms to the search engine. To our knowledge such situations are not handled by any current query model estimation methods in a principled way.

To remedy these problems, we need a better theoretical framework for query model estimation: one that incorporates both risk and reward data about terms, that detect risky situations and expands selectively, that can incorporate arbitrary additional problem constraints such as a computational budget, and has fast practical implementations.

Our solution is to develop a novel formulation of query model estimation as a convex optimization problem [1], by casting the problem in terms of constrained graph labeling. Informally, we seek query models that use a set of terms with high expected relevance but low expected risk. This idea has close connections with models of risk in portfolio optimization [7]. An optimization approach frees us from the need to provide a closed-form formula for term weighting. Instead, we specify a (convex) objective function and a set of constraints that a good query model should satisfy, letting the solver do the work of searching the space of feasible query models. This approach gives a natural way to perform selective expansion: if there is no feasible solution to the optimization problem, we do not attempt to expand the original query. ore generally, it gives a very flexible framework for integrating different criteria for expansion as optimization constraints or objectives.

Our risk framework consists of two key parts. First, we seek to minimize an objective function that consists of two criteria: term relevance, and term risk. Term risk in turn has two subcomponents: the *individual risk* of a term, and the *conditional risk* of choosing one term given we have already chosen another. Second, we specify constraints on what 'good' sets of terms should look like. These constraints are chosen to address traditional reasons for query drift. With these two parts, we obtain a simple convex program for solving for the relative term weights in a query model.

## 2 Theoretical model

Our aim in this section is to develop a constrained optimization program to find stable, effective query models. Typically, our optimization will embody a basic tradeoff between wanting to use evidence that has strong expected relevance, such as expansion terms with high relevance model weights, and the risk or confidence in using that evidence. We begin by describing the objectives and constraints over term sets that might be of interest for estimating query models. We then describe a set of (sometimes competing) constraints whose feasible set reflects query models that are likely to be effective and reliable. Finally, we put all these together to form the convex optimization problem.

### 2.1 Query model estimation as graph labeling

We can gain some insight into the problem of query model estimation by viewing the process of building a query as a two-class *labeling* problem over terms. Given a vocabulary $V$, for each term $t \in V$ we decide to either add term $t$ to the query (assign label '1' to the term), or to leave it out (assign label '0'). The initial query terms are given a label of '1'. Our goal is to find a function $f : V \rightarrow \{0, 1\}$ that classifies the finite set $V$ of $|V| = K$ terms, choosing one of the two labels for each term. The terms are typically related, so that the pairwise similarity $\sigma(i, j)$ between any two terms $w_i, w_j$ is represented by the weight of the edge connecting $w_i$ and $w_j$ in the undirected graph $G = (V, E)$, where $E$ is the set of all edges. The cost function $L(f)$ captures our displeasure for a given $f$, according to how badly the following two criteria are given by the labeling produced by $f$.

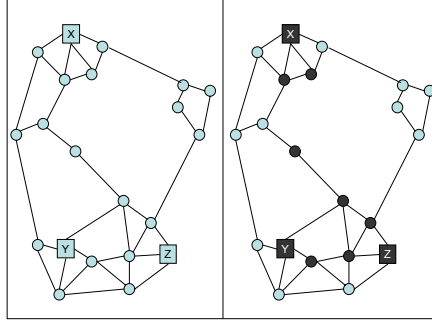

Figure 1: Query model estimation as a constrained graph labeling problem using two labels (relevant, non-relevant) on a graph of pairwise term relations. The square nodes X, Y, and Z represent query terms, and circular nodes represent potential expansion terms. Dark nodes represent terms with high estimated label weights that are likely to be added to the initial query. Additional constraints can select sets of terms having desirable properties for stable expansion, such as a bias toward relevant labels related to multiple query terms (right).

- The cost $c_{i:k}$ gives the cost of labeling term $t_i$ with label $k \in \{0, 1\}$.

- The cost $\sigma_{i,j} \cdot d(f(i), f(j))$ gives the penalty for assigning labels $f(i)$ and $f(j)$ to items $i$ and $j$ when their similarity is $\sigma_{i,j}$. The function $d(u, v)$ is a metric that is the same for all edges. Typically, similar items are expected to have similar labels and thus a penalty is assigned to the degree this expectation is violated.

For this study, we assume a very simple metric in which $d(i, j) = 1$ if $i \neq j$ and $0$ otherwise. In a probabilistic setting, finding the most probable labeling can be viewed as a form of maximum a posteriori (MAP) estimation over the Markov random field defined by the term graph.

Although this problem is NP-hard for arbitrary configurations, various approximation algorithms exist that run in polynomial time by relaxing the constraints. Here we relax the condition that the labels be integers in $\{0, 1\}$ and allow real values in $[0, 1]$. A review of relaxations for the more general metric labeling problem is given by Ravikumar and Lafferty [10]. The basic relaxation we use is

$$\text{maximize} \quad \sum_{s;j} c_{s;j} x_{s;j} + \sum_{s,t;j,k} \sigma_{s,j;t,k} x_{s;j} x_{t;k}$$

$$\text{subject to} \quad \sum_{j} x_{s;j} = 1 \tag{1}$$

$$0 \leq x_{s;j} \leq 1.$$

The variable $x_{s;j}$ denotes the assignment value of label $j$ for term $s$. Our method obtains its initial assignment costs $c_{s;j}$ from a baseline feedback method, given an observed query and corresponding set of query-ranked documents. For our baseline expansion method, we use the strong default feedback algorithm included in Indri 2.2 based on Lavrenko's Relevance Model [5]. Further details are available in [4].

In the next section, we discuss how to specify values for $c_{s;j}$ and $\sigma_{s,j;t,k}$ that make sense for query model estimation. For a two-label problem where $j \in \{0, 1\}$, the values of $x_i$ for one label completely determine the values for the other, since they must sum to 1, so it suffices to optimize over only the $x_{i;1}$, and for simplicity we simply refer to $x_i$ instead of $x_{i;1}$.

Our goal is to find a set of weights $x = (x_1, \ldots, x_K)$ where each $x_i$ corresponds to the weight in the final query model of term $w_i$ and thus is the relative value of each word in the expanded query. The graph labeling formulation may be interpreted as combining two natural objectives: the first maximizes the expected relevance of the selected terms, and the second minimizes the risk associated with the selection. We now describe each of these in more detail, followed by a description of additional set-based constraints that are useful for query expansion.

## 2.2 Relevance objectives

Given an initial set of term weights from a baseline expansion method $c = (c_1, \ldots, c_K)$ the *expected relevance* over the vocabulary $V$ of a solution $x$ is given by the weighted sum $c \cdot x = \sum_k c_k x_k$. Essentially, maximizing expected relevance biases the 'relevant' labels toward those words with the highest $c_i$ values. Other relevance objective functions are also possible, as long as they are convex. For example, if $c$ and $x$ represent probability distributions over terms, then we could replace $c \cdot x$ with $KL(c||x)$ as an objective since KL-divergence is also convex in $c$ and $x$.

The initial assignment costs (label values) $c$ can be set using a number of methods depending on how scores from the baseline expansion model are normalized. In the case of Indri's language model-based expansion, we are given estimates of the Relevance Model $p(w|R)$ over the highest-ranking $k$ documents[1]. We can also estimate a non-relevance model $p(w|N)$ using the collection to approximate non-relevant documents, or using the *lowest-ranked* $k$ documents out of the top 1000 retrieved by the initial query $Q$. To set $c_{s:1}$, we first compute $p(R \mid w)$ for each word $w$ via Bayes Theorem,

$$p(R|w) = \frac{p(w|R)}{p(w|R) + p(w|N)} \tag{2}$$

assuming $p(R) = p(N) = 1/2$. Using the notation $p(R|Q)$ and $p(R|\bar{Q})$ to denote our belief that any query word or non-query word respectively should have label 1, the initial expected label value is then

$$c_{s:1} = \begin{cases} p(R|Q) + (1 - p(R|Q)) \cdot p(R|w_s) & s \in Q \\ p(R|\bar{Q}) \cdot p(R|w_s) & s \notin Q \end{cases} \tag{3}$$

for the 'relevant' label. We use $p(R|Q) = 0.75$ and $p(R|\bar{Q}) = 0.5$. Since the label values must sum to one, for binary labels we have $c_{s:0} = 1 - c_{s:1}$.

## 2.3 Risk objectives

Optimizing for expected term relevance only considers one dimension of the problem. A second critical objective is minimizing the risk associated with a particular term labeling. We adapt an informal definition of risk here in which the variance of the expected relevance is a proxy for uncertainty, encoded in the matrix $\Sigma$ with entries $\sigma_{ij}$. Using a betting analogy, the weights $x = \{x_i\}$ represent wagers on the utility of the query model terms. A risky strategy would place all bets on the single term with highest relevance score. A lower-risk strategy would distribute bets among terms that had both a large estimated relevance and low redundancy, to cover all aspects of the query.

**Conditional term risk.** First, we consider the *conditional risk* $\sigma_{ij}$ between pairs of terms $w_i$ and $w_j$. To quantify conditional risk, we measure the redundancy of choosing word $w_i$ given that $w_j$ has already been selected. This relation is expressed by choosing a symmetric similarity measure $\sigma(w_i, w_j)$ between $w_i$ and $w_j$, which is rescaled into a distance-like measure $d(w_i, w_j)$ with the formula

$$\sigma_{ij} = d(w_i, w_j) = \gamma \exp(-\rho \cdot \sigma(w_i, w_j)) \tag{4}$$

The quantities $\gamma$ and $\rho$ are scaling constants that depend on the output scale of $\sigma$, and the choice of $\gamma$ also controls the relative importance of individual vs. conditional term risk. In this study, our $\sigma(w_i, w_j)$ measure is based on term associations over the $2 \times 2$ contingency table of term document counts. For this experiment we used the Jaccard coefficient: future work will examine others.

**Individual risk.** We say that a term related to multiple query terms exhibits *term centrality*. Previous work has shown that central terms are more likely to be more effective for expansion than terms related to few query terms [3] [12]. We use term centrality to quantify a term's individual risk, and define it for a term $w_i$ in terms of the vector $d_i$ of all similarities of $w_i$ with all query terms. The covariance matrix $\Sigma$ then has diagonal entries

$$\sigma_{ii} = \|d_i\|_2^2 = \sum_{w_q \in Q} d^2(w_i, w_q) \tag{5}$$

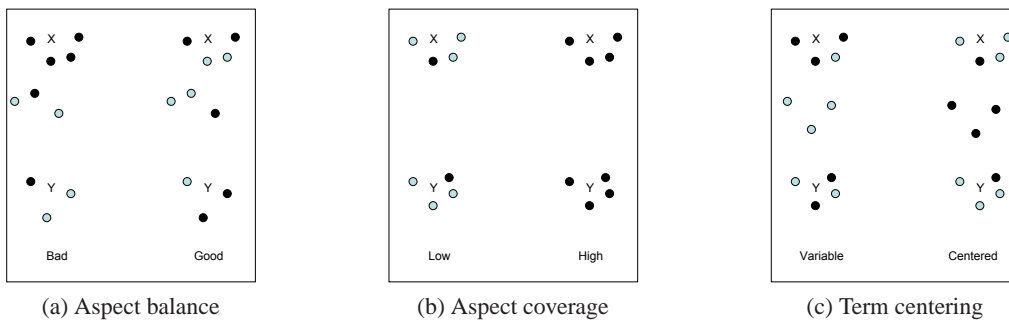

(a) Aspect balance          (b) Aspect coverage          (c) Term centering

Figure 2: Three complementary criteria for expansion term weighting on a graph of candidate terms, and two query terms $X$ and $Y$. The aspect balance constraint (left) prefers sets of expansion terms that balance the representation of $X$ and $Y$. The aspect coverage constraint (center) increases recall by allowing more expansion candidates within a distance threshold of each term. Term centering (right) prefers terms near the center of the graph, and thus more likely to be related to both terms, with minimum variation in the distances to $X$ and $Y$.

Other definitions of centrality are certainly possible, e.g. depending on generative assumptions for term distributions.

We can now combine relevance and risk into a single objective, and control the tradeoff with a single parameter $\kappa$, by minimizing the function

$$L(x) = -c^T x + \frac{\kappa}{2} x^T \Sigma x. \tag{6}$$

If $\Sigma$ is estimated from term co-occurrence data in the top-retrieved documents, then the condition to minimize $x^T \Sigma x$ also encodes the fact that we want to select expansion terms that are not all in the same co-occurrence cluster. Rather, we prefer a set of expansion terms that are more diverse, covering a larger range of potential topics.

## 2.4 Set-based constraints

One limitation of current query model estimation methods is that they typically make greedy term-by-term decisions using a threshold, without considering the qualities of the set of terms as a whole. A one-dimensional greedy selection by term score, especially for a small number of terms, has the risk of emphasizing terms related to one aspect and not others. This in turn increases the risk of query drift after expansion. We now define several useful constraints on query model terms: *aspect balance*, *aspect coverage*, and *query term support*. Figure 2 gives graphical examples of aspect balance, aspect coverage, and the term centrality objective.

**Aspect balance.**  We make the simplistic assumption that each of a query's terms represents a separate and unique aspect of the user's information need. We create the matrix $A$ from the vectors $\phi_k(w_i)$ for each query term $q_k$, by setting $A_{ki} = \phi_k(w_i) = \sigma_{ik}$. In effect, $Ax$ gives the projection of the solution model $x$ on each query term's feature vector $\phi_k$. We define the requirement that $x$ be in balance to be that the vector $Ax$ be element-wise close to the mean vector $\mu$ of the $\phi_k$, within a tolerance $\zeta_\mu$, which we denote (with some flexibility in notation) by

$$Ax \preceq \mu + \zeta_\mu. \tag{7}$$

To demand an exact solution, we set $\zeta_\mu = 0$. In reality, some slack is desirable for slightly better results and so we use a small positive value for $\zeta_\mu$ such as $1.0$.

**Query term support.**  Another important constraint is that the set of initial query terms $Q$ be predicted by the solution labeling. We express this mathematically by requiring that the the weights for the 'relevant' label on the query terms $x_{i:1}$ lie in a range $l_i \leq x_i \leq u_i$ and in particular be above the threshold $l_i$ for $x_i \in Q$. Currently $l_i$ is set to a default value of $0.95$ for all query terms, and zero for all other terms. $u_i$ is set to $1.0$ for all terms. Term-specific values for $l_i$ may also be desirable to reflect the rarity or ambiguity of individual query terms.

$$\text{minimize} \quad -c^T x + \frac{\kappa}{2} x^T \Sigma x \qquad \qquad \text{\textit{Relevance, term centrality \& risk}} \qquad (9)$$

$$\text{subject to} \quad Ax \preceq \mu + \zeta_\mu \qquad \qquad \qquad \text{\textit{Aspect balance}} \qquad (10)$$

$$g_i^{\ T} x \geq \zeta_i, \qquad w_i \in Q \qquad \qquad \text{\textit{Aspect coverage}} \qquad (11)$$

$$l_i \leq x_i \leq u_i, \quad i = 1, \dots, K \qquad \text{\textit{Query term support, positivity}} \qquad (12)$$

Figure 3: The basic constrained quadratic program QMOD used for query model estimation.

**Aspect coverage.**  One of the strengths of query expansion is its potential for solving the vocabulary mismatch problem by finding different words to express the same information need. Therefore, we can also require a minimal level of *aspect coverage*. That is, we may require more than just that terms are balanced evenly among all query terms: we may care about the absolute level of support that exists. For example, suppose our information sources are feedback terms, and we have two possible term weightings that are otherwise feasible solutions. The first weighting has only enough terms selected to give a minimal non-zero but even covering to all aspects. The second weighting scheme has three times as many terms, but also gives an even covering. Assuming no conflicting constraints such as maximum query length, we may prefer the second weighting because it increases the chance we find the right alternate words for the query, potentially improving recall.

We denote the set of distances to neighboring words of query term $q_i$ by the vector $g_i$. The projection $g_i^{\ T} x$ gives us the aspect coverage, or how well the words selected by the solution $x$ 'cover' term $q_i$. The more expansion terms near $q_i$ that are given higher weights, the larger this value becomes. When only the query term is covered, the value of $g_i^{\ T} x = \sigma_{ii}$. We want the aspect coverage for each of the vectors $g_i$ to exceed a threshold $\zeta_i$, and this is expressed by the constraint

$$g_i^{\ T} x \geq \zeta_i. \qquad (8)$$

Putting together the relevance and risk objectives, and constraining by the set properties, results in the following complete quadratic program for query model estimation, which we call QMOD and is shown in Figure 3. The role of each constraint is given in italics.

## 3  Evaluation

In this section we summarize the effectiveness of using the QMOD convex programs to estimate query models and examine how well the QMOD feasible set is calibrated to the empirical risk of expansion. For space reasons we are unable to include a complete sensitivity analysis of the effect of the various constraints. The best risk-reward tradeoff is generally obtained with a strong query support constraint ($l_i$ near 1.0) and moderate balance between individual and conditional term risk. We used the following default values for the control parameters: $\kappa = 1.0$, $\gamma = 0.75$, $\zeta_\mu = 1.0$, $\zeta_i = 0.1$, $u_i = 1.0$, and $l_i = 0.95$ for query terms and $l_i = 0$ for non-query terms.

### 3.1  Robustness of Model Estimation

In this section we evaluate the robustness of the query models estimated using the convex program in Fig. 3 over several TREC collections. We created a histogram of MAP improvement across sets of topics. This is a fine-grained look that shows the distribution of gain or loss in MAP for a given feedback method. Using these histograms we can distinguish between two systems that might have the same number of failures, but which help or hurt queries by very different magnitudes. The number of queries helped or hurt by expansion is shown, binned by the loss or gain in average precision by using feedback. The baseline feedback here was Indri 2.2 (Modified Relevance Model with stoplist) [8]. The robustness histogram with results combined for all collections is shown in Fig. 4. Both algorithms achieve the same gain in average precision over all collections (15%). Yet considering the expansion failures whose loss in average precision is more than 10%, the robust version hurts more than 60% fewer queries.

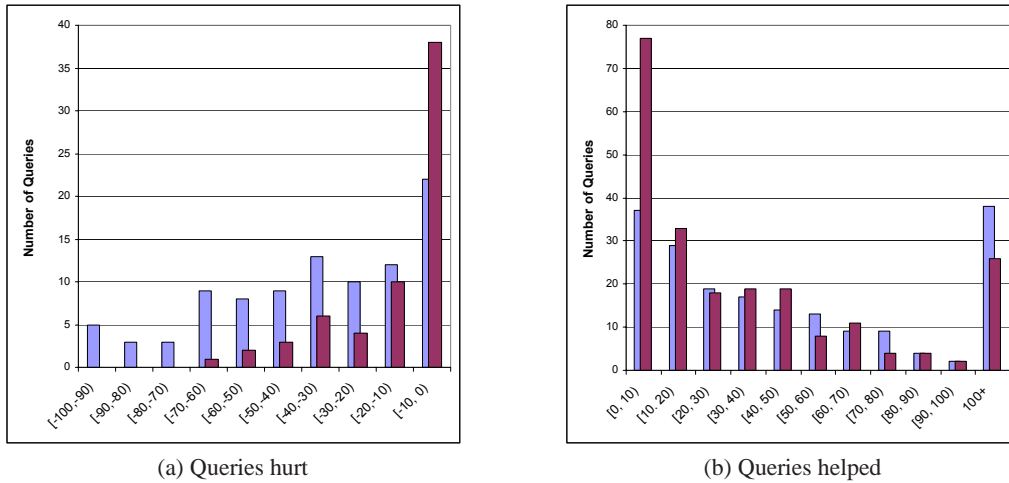

| (a) Queries hurt | (b) Queries helped |

Figure 4: Comparison of expansion robustness for four TREC collections combined (TREC 1&2, TREC 7, TREC 8, wt10g). The histograms show counts of queries, binned by percent change in average precision. The dark bars show robust expansion performance using the QMOD convex program with default control parameters. The light bars show baseline expansion performance using term relevance weights only. Both methods improve average precision by an average of 15%, but the robust version hurts significantly fewer queries, as evident by the greatly reduced tail on the left histogram (queries hurt).

## 3.2   Calibration of Feasible Set

If the constraints of a convex program are well-designed for stable query expansion, the odds of an infeasible solution should be much greater than 50% for queries that are risky. In those cases, the algorithm will not attempt to enhance the query. Conversely, the odds of finding a feasible query model should ideally increase for thoese queries that are more amenable to expansion. Overall, 17% of all queries had infeasible programs. We binned these queries according to the actual gain or loss that would have been achieved with the baseline expansion, normalized by the original number of queries appearing in each bin when the (non-selective) baseline expansion is used. This gives the log-odds of reverting to the original query for any given gain/loss level.

The results are shown in in Figure 5. As predicted, the QMOD algorithm is more likely to decide infeasibility for the high-risk zones at the extreme ends of the scale. Furthermore, the odds of finding a feasible solution do indeed increase directly with the actual benefits of using expansion, up to a point where we reach an average precision gain of 75% and higher. At this point, such high-reward queries are considered high risk by the algorithm, and the likelihood of reverting to the original query increases dramatically again. This analysis makes clear that the selective expansion behavior of the convex algorithm is well-calibrated to the true expansion benefit.

## 4   Conclusions

We have presented a new research approach to query model estimation, showing how to adapt convex optimization methods to the problem by casting it as constrained graph labeling. By integrating relevance and risk objectives with additional constraints to selectively reduce expansion for the most risky queries, our approach is able to significantly reduce the downside risk of a strong baseline algorithm while retaining its strong gains in average precision.

Our expansion framework is quite general and easily accomodates further extensions and refinements. For example, similar to methods used for portfolio optimization [6] we can assign a computational cost to each term having non-zero weight, and add budget constraints to prefer more efficient expansions. In addition, sensitivity analysis of the constraints is likely provide useful information for active learning: interesting extensions to semi-supervised learning are possible to incorporate additional observations such as relevance feedback from the user. Finally, there are a number of

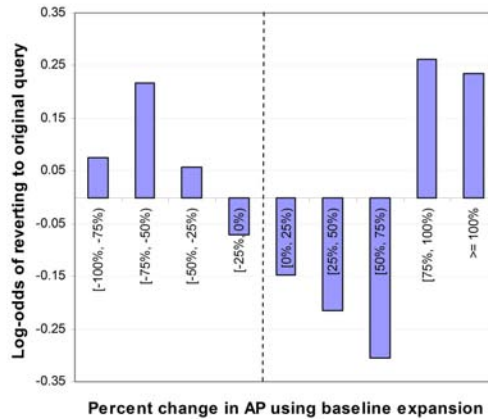

Figure 5: The log-odds of reverting to the original query as a result of selective expansion. Queries are binned by the percent change in average precision if baseline expansion were used. Columns above the line indicate greater-than-even odds that we revert to the original query.

higher-level control parameters and it would be interesting to determine the optimal settings. The values we use have not been extensively tuned, so that further performance gains may be possible.

**Acknowledgments**

We thank Jamie Callan, John Lafferty, William Cohen, and Susan Dumais for their valuable feedback on many aspects of this work.

## Footnotes

[1]We use the symbols $R$ and $N$ to represent relevance and non-relevance respectively.

# References

[1] S. Boyd and L. Vandenberghe. *Convex Optimization*. Cambridge University Press, 2004.

[2] C. Buckley. Why current IR engines fail. In *Proceedings of the 27th Annual International ACM SIGIR Conference on Research and Development in Information Retrieval (SIGIR 2004)*, pages 584–585, 2004.

[3] K. Collins-Thompson and J. Callan. Query expansion using random walk models. In *Proc. of the 14th International Conf. on Information and Knowledge Management (CIKM 2005)*, pages 704–711, 2005.

[4] K. Collins-Thompson and J. Callan. Estimation and use of uncertainty in pseudo-relevance feedback. In *Proceedings of the 30th Annual International ACM SIGIR Conference on Research and Development in Information Retrieval (SIGIR 2007)*, pages 303–310, 2007.

[5] V. Lavrenko. *A Generative Theory of Relevance*. PhD thesis, Univ. of Massachusetts, Amherst, 2004.

[6] M. S. Lobo, M. Fazel, and S. Boyd. Portfolio optimization with linear and fixed transaction costs. *Annals of Operations Research*, 152(1):376–394, 2007.

[7] H. M. Markowitz. Portfolio selection. *Journal of Finance*, 7(1):77–91, 1952.

[8] D. Metzler and W. B. Croft. Combining the language model and inference network approaches to retrieval. *Information Processing and Management*, 40(5):735–750, 2004.

[9] J. M. Ponte and W. B. Croft. A language modeling approach to information retrieval. In *Proc. of the 1998 ACM SIGIR Conference on Research and Development in Information Retrieval*, pages 275–281, 1998.

[10] P. Ravikumar and J. Lafferty. Quadratic programming relaxations for metric labeling and markov random field map estimation. In *Proceedings of the 23rd International Conference on Machine Learning (ICML 2006)*, pages 737–744, 2006.

[11] J. Teevan, S. T. Dumais, and E. Horvitz. Personalizing search via automated analysis of interests and activities. In *Proceedings of the 28th Annual International ACM SIGIR Conference on Research and Development in Information Retrieval (SIGIR 2005)*, pages 449–456, New York, NY, USA, 2005. ACM.

[12] J. Xu and W. B. Croft. Query expansion using local and global document analysis. In *Proceedings of the 1996 Annual International ACM SIGIR Conference on Research and Development in Information Retrieval*, pages 4–11, 1996.
